# Predictive Subspace Learning for Multi-view Data: a Large Margin Approach

**Ning Chen**[†‡]          **Jun Zhu**[‡]          **Eric P. Xing**[‡]

[†]chenn07@mails.tsinghua.edu.cn, [‡]{ningchen,junzhu,epxing}@cs.cmu.edu

[†]Dept. of CS & T, TNList Lab, State Key Lab of ITS, Tsinghua University, Beijing 100084 China

[‡]School of Computer Science, Carnegie Mellon University, Pittsburgh, PA 15213 USA

## Abstract

Learning from multi-view data is important in many applications, such as image classification and annotation. In this paper, we present a large-margin learning framework to discover a predictive latent subspace representation shared by multiple views. Our approach is based on an undirected latent space Markov network that fulfills a weak conditional independence assumption that multi-view observations and response variables are independent given a set of latent variables. We provide efficient inference and parameter estimation methods for the latent subspace model. Finally, we demonstrate the advantages of large-margin learning on real video and web image data for discovering predictive latent representations and improving the performance on image classification, annotation and retrieval.

## 1   Introduction

In many scientific and engineering applications, such as image annotation [28] and web-page classification [6], the available data usually come from diverse domains or are extracted from different aspects, which will be referred to as *views*. Standard predictive methods, such as support vector machines, are built with all the variables available, without taking into consideration the presence of distinct views. These methods would sacrifice the predictive performance [7] and may also be incapable of performing *view-level analysis* [12], such as predicting the tags for image annotation and analyzing the underlying relationships amongst views. Different from the existing work that has been done on exploring multi-view information to alleviate the difficult semi-supervised learning [6, 12, 2, 14] and unsupervised clustering [8] problems, our goal is to develop a statistical framework that learns a *predictive* subspace representation shared by multiple views when labels are provided and perform view-level analysis, particularly view-level predictions.

To discover a subspace representation shared by multi-view data, the unsupervised canonical correlation analysis (CCA) [17] and its kernelized version [1] ignore the widely available supervised information, such as image categories. Therefore, they could discover a subspace with weak predictive ability. The multi-view fisher discriminant analysis (FDA) [13] provides a supervised approach to finding such a projected subspace. However, this deterministic approach cannot provide view-level predictions, such as image annotation; and it would also need a density estimator in order to apply the information criterion [9] to detect view disagreement. In this paper, we consider a probabilistic approach to model multi-view data, which can perform both the response-level predictions (e.g., image classification) and view-level predictions (e.g., image annotation).

Specifically, we propose a large-margin learning approach to discovering a predictive subspace representation for multi-view data. The approach is based on a generic *multi-view latent space Markov network* (MN) that fulfills a weak conditional independence assumption that the data from different views and the response variables are conditionally independent given a set of latent variables. This conditional independence is much weaker than the typical assumption (e.g., in the seminal work of

co-training [6]) that multi-view data are conditionally independent given the very low dimensional response variables [14]. Although directed Bayesian networks (BNs) (e.g., latent Dirichlet allocation (LDA) [5] and probabilistic CCA [3]) can also be designed to fulfill the conditional independence, the posterior inference can be hard because all the latent variables are coupled together given the input variables [26]. Therefore, we ground our approach on the undirected MNs. Undirected latent variable models have shown promising performance in many applications [26, 20]. In the multi-view MN, conditioned on latent variables, each view defines a joint distribution similar to that in a conditional random field (CRF) [18] and thus it can effectively extract latent topics from structured data. For example, considering word ordering information could improve the quality of discovered latent topics [23] compared to a method (e.g., LDA) solely based on the natural bag-of-word representation, and spatial relationship among regions in an image is also useful for computer vision applications [15]. To learn the multi-view latent space MN, we develop a large-margin approach, which jointly maximizes the data likelihood and minimizes the hinge-loss on training data. The learning and inference problems are efficiently solved with a contrastive divergence method [25]. Finally, we concentrate on one special case of the large-margin mult-view MN and extensively evaluate it on real video and web image datasets for image classification, annotation and retrieval tasks. Our results show that the large-margin approach can achieve significant improvements in terms of prediction performance and discovered latent subspace representations.

The paper is structured as follows. Sec 2 and Sec 3 present the multi-view latent space MN and its large-margin training. Sec 4 presents a special case. Sec 5 presents empirical results and Sec 6 concludes.

## 2 Multi-view Latent Space Markov Networks

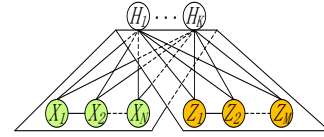

The unsupervised two-view latent space Markov network is shown in Fig. 1, which consists of two views of input data $\mathbf{X} := \{X_n\}$ and $\mathbf{Z} := \{Z_m\}$ and a set of latent variables $\mathbf{H} := \{H_k\}$. For ease of presentation, we assume that the variables on each view are connected via a linear-chain. Extensions to multiple views and

Figure 1: Multi-view Markov networks with $K$ latent variables.

more complex structures on each view can be easily done, after we have presented the constructive definition of the model distribution. The model is constructed based on an underlying conditional independence assumption that given the latent variables $\mathbf{H}$, the two views $\mathbf{X}$ and $\mathbf{Z}$ are independent.

Graphically, we can see that both the exponential family Harmonium (EFH) [26] and its extension of dual-wing Harmonium (DWH) [28] are special cases of multi-view latent space MNs. Therefore, it is not surprising to see that multi-view MNs inherit the widely advocated property of EFH that the model distribution can be constructively defined based on local conditionals on each view. Specifically, we first define marginal distributions of the data on each view and the latent variables. For each view, we consider the first-order Markov network. By the random field theory, we have

$$p(\mathbf{x}) = \exp\left\{\sum_i \theta_i^\top \phi(x_i, x_{i+1}) - A(\theta)\right\}, \text{ and } p(\mathbf{z}) = \exp\left\{\sum_j \eta_j^\top \psi(z_j, z_{j+1}) - B(\eta)\right\},$$

where $\phi$ and $\psi$ are feature functions, $A$ and $B$ are log partition functions. For latent variables $\mathbf{H}$, each component $h_k$ has an exponential family distribution and therefore the marginal distribution is:

$$p(\mathbf{h}) = \prod_k p(h_k) = \prod_k \exp\left\{\lambda_k^\top \varphi(h_k) - C_k(\lambda_k)\right\},$$

where $\varphi(h_k)$ is the feature vector of $h_k$, $C_k$ is another log-partition function.

Next, the joint model distribution is defined by combining the above components in the log-domain and introducing additional terms that couple the random variables $\mathbf{X}$, $\mathbf{Z}$ and $\mathbf{H}$. Specifically, we have

$$p(\mathbf{x}, \mathbf{z}, \mathbf{h}) \propto \exp\left\{\sum_i \theta_i^\top \phi(x_i, x_{i+1}) + \sum_j \eta_j^\top \psi(z_j, z_{j+1}) + \sum_k \lambda_k^\top \varphi(h_k) \right.$$
$$\left. + \sum_{ik} \phi(x_i, x_{i+1})^\top \mathbf{W}_i^k \varphi(h_k) + \sum_{jk} \psi(z_j, z_{j+1})^\top \mathbf{U}_j^k \varphi(h_k)\right\}. \tag{1}$$

Then, we can directly write the conditional distributions on each view with shifted parameters,

$$p(\mathbf{x}|\mathbf{h}) = \exp\left\{\sum_i \hat{\theta}_i^\top \phi(x_i, x_{i+1}) - A(\hat{\theta})\right\}, \text{ where } \hat{\theta}_i = \theta_i + \sum_k \mathbf{W}_i^k \varphi(h_k);$$

$$p(\mathbf{z}|\mathbf{h}) = \exp\left\{\sum_j \hat{\eta}_j^\top \psi(z_j, z_{j+1}) - B(\hat{\eta})\right\}, \text{ where } \hat{\eta}_j = \eta_j + \sum_k \mathbf{U}_j^k \varphi(h_k); \text{ and}$$

$$p(\mathbf{h}|\mathbf{x}, \mathbf{z}) = \prod_k \exp\left\{\hat{\lambda}_k^\top \varphi(h_k) - C_k(\hat{\lambda}_k)\right\}, \text{ where } \hat{\lambda}_k = \lambda_k + \sum_i \mathbf{W}_i^k \phi(x_i, x_{i+1}) + \sum_j \mathbf{U}_j^k \psi(z_j, z_{j+1}).$$

We can see that conditioned on the latent variables, both $p(\mathbf{x}|\mathbf{h})$ and $p(\mathbf{z}|\mathbf{h})$ are defined in the exponential form with a pairwise potential function, which is very similar to conditional random fields [18]. Reversely, we can start with defining the local conditional distributions as above and directly write the compatible joint distribution, which is of the log-linear form as in (1). We will use $\Theta$ to denote all the parameters $(\theta, \eta, \lambda, \mathbf{W}, \mathbf{U})$.

Since the latent variables are not directly connected, the complexity of inferring the posterior distribution of $\mathbf{H}$ is the same as in EFH when all the input data are observed, as reflected in the factorized form of $p(\mathbf{h}|\mathbf{x}, \mathbf{z})$. Therefore, multi-view latent space MNs do not increase the complexity on testing if our task depends solely on the latent representation (i.e., expectation of $\mathbf{H}$), such as information retrieval [26], classification, clustering etc. However, the complexity of parameter estimation and inferring the posterior distribution of each view (e.g., $\mathbf{X}$) will be increased, depending on the structure on the view. For the simple case of linear-chain, the inference can be efficiently done with a forward-backward message passing scheme [18]. For a general model structure, which may contain many loops, approximate inference such as variational methods [22] is needed to perform the task. We will provide more details when presenting the learning problem.

Up to now, we have sticken on unsupervised multi-view latent space MNs, which are of wide use in discovering latent subspace representations shared by multi-view data. In this paper, however, we are more interested in the supervised setting where each input sample is associated with a supervised response variable, such as image categories. Accordingly, our goal is to discover a *predictive* subspace by exploring the supervised information. The supervised multi-view latent space MNs are defined similarly as above, but with an additional view of response variables $Y$. Now, the conditional independence is: $\mathbf{X}$, $\mathbf{Z}$ and $Y$ are independent if $\mathbf{H}$ is given. As we have stated, this assumption is much weaker than the typical conditional independence assumption that $\mathbf{X}$ and $\mathbf{Z}$ are independent given $Y$. Based on the constructive definition, we only need to specify the conditional distribution of $Y$ given $\mathbf{H}$. In principle, $Y$ can be continuous or discrete. Here, we consider the discrete case, where $y \in \{1, \cdots, T\}$, and define

$$p(y|\mathbf{h}) = \frac{\exp\{\mathbf{V}^\top \mathbf{f}(\mathbf{h}, y)\}}{\sum_{y'} \exp\{\mathbf{V}^\top \mathbf{f}(\mathbf{h}, y')\}}, \tag{2}$$

where $\mathbf{f}(\mathbf{h}, y)$ is the feature vector whose elements from $(y-1)K + 1$ to $yK$ are those of $\mathbf{h}$ and all others are 0. Accordingly, $\mathbf{V}$ is a stacking parameter vector of $T$ sub-vectors $\mathbf{V}_y$, of which each one corresponds to a class label $y$. Then, the joint distribution $p(\mathbf{x}, \mathbf{z}, \mathbf{h}, y)$ has the same form as in Eq. (1), but with an additional term of $\mathbf{V}^\top \mathbf{f}(\mathbf{h}, y) = \mathbf{V}_y^\top \mathbf{h}$ in the exponential.

We note that a supervised version of DWH, which will be denoted by TWH (i.e., triple wing Harmonium), was proposed in [29], and the parameter estimation was done by maximizing the joint data likelihood. However, the resultant TWH model does not yield improved performance compared to the naive method that combines an unsupervised DWH for discovering latent representations and an SVM for classification. This observation further motivates us to develop a more discriminative learning approach to exploring the supervised information for discovering predictive latent subspace representations. As we shall see, integrating the large-margin principle into one objective function for joint latent subspace model and prediction model learning can yield much better results, in terms of prediction performance and predictiveness of discovered latent subspace representations.

## 3   Parameter Estimation: a Large Margin Approach

To learn the supervised multi-view latent space MNs, a natural method is the maximum likelihood estimation (MLE), which has been widely used to train directed [24, 30] and undirected latent variable models [26, 20, 28, 29]. However, likelihood-based parameter estimation pays additional efforts in defining a normalized probabilistic model as in Eq. (2), of which the normalization factor can make the inference hard, especially in directed models [24]. Moreover, the standard MLE could result in non-conclusive results, as reported in [29] and verified in our experiments. These have been motivating us to develop a more discriminative learning approach. An arguably more discriminative way to learn a classification model is to directly estimate the decision boundary, which is the essential idea underlying the very successful large-margin classifiers (e.g., SVMs). Here, we integrate the large-margin idea into the learning of supervised multi-view latent space MNs for multi-view data analysis, analogous to the development of MedLDA [31], which is directed and has single-view. For brevity, we consider the general multi-class classification, as defined above.

### 3.1 Problem Definition

As in the log-linear model in Eq. (2), we assume that the discriminant function $F(y, \mathbf{h}; \mathbf{V})$ is linear, that is, $F(y, \mathbf{h}; \mathbf{V}) = \mathbf{V}^\top \mathbf{f}(\mathbf{h}, y)$, where $\mathbf{f}$ and $\mathbf{V}$ are defined the same as above. For prediction, we take the expectation over the latent variable $\mathbf{H}$ and define the prediction rule as

$$y^* := \arg\max_y \mathbb{E}_{p(\mathbf{h}|\mathbf{x},\mathbf{z})}[F(\mathbf{H}, y; \mathbf{V})] = \arg\max_y \mathbf{V}^\top \mathbb{E}_{p(\mathbf{h}|\mathbf{x},\mathbf{z})}[\mathbf{f}(\mathbf{H}, y)], \qquad (3)$$

where the expectation can be efficiently computed with the factorized form of $p(\mathbf{h}|\mathbf{x}, \mathbf{z})$ when $\mathbf{x}$ and $\mathbf{z}$ are fully observed. If missing values exist in $\mathbf{x}$ or $\mathbf{z}$, an inference procedure is needed to compute the expectation of the missed components, as detailed below in Eq. (5).

Then, learning is to find an optimal $\mathbf{V}^*$ that minimizes a loss function. Here, we minimize the hinge loss, as used in SVMs. Given training data $\mathcal{D} = \{(\mathbf{x}_d, \mathbf{z}_d, y_d)\}_{d=1}^D$, the hinge loss of the predictive rule (3) is

$$\mathcal{R}_{hinge}(\mathbf{V}) := \frac{1}{D} \sum_d \max_y [\Delta\ell_d(y) - \mathbf{V}^\top \mathbb{E}_{p(\mathbf{h}|\mathbf{x},\mathbf{z})}[\Delta\mathbf{f}_d(y)]],$$

where $\Delta\ell_d(y)$ is a loss function that measures how different the prediction $y$ is compared to the true label $y_d$, and $\mathbb{E}_{p(\mathbf{h}|\mathbf{x},\mathbf{z})}[\Delta\mathbf{f}_d(y)] = \mathbb{E}_{p(\mathbf{h}|\mathbf{x},\mathbf{z})}[\mathbf{f}(\mathbf{H}_d, y_d)] - \mathbb{E}_{p(\mathbf{h}|\mathbf{x},\mathbf{z})}[\mathbf{f}(\mathbf{H}_d, y)]$. It can be proved that the hinge loss is an upper bound of the empirical loss $\mathcal{R}_{emp} := \frac{1}{D} \sum_d \Delta\ell(y_d^*)$. Applying the principle of *regularized risk minimization*, we define the learning problem as solving

$$\min_{\Theta, \mathbf{V}} \ L(\Theta) + \frac{1}{2} C_1 \|\mathbf{V}\|_2^2 + C_2 \mathcal{R}_{hinge}(\mathbf{V}), \qquad (4)$$

where $L(\Theta) := -\sum_d \log p(\mathbf{x}_d, \mathbf{z}_d)$ is the negative data likelihood and $C_1$ and $C_2$ are non-negative constants, which can be selected via cross-validation. Note that $\mathcal{R}_{hinge}$ is also a function of $\Theta$.

Since problem (4) jointly maximizes the data likelihood and minimizes a training loss, it can be expected that by solving this problem we can find a predictive latent space representation $p(\mathbf{h}|\mathbf{x}, \mathbf{z})$ and a prediction model parameter $\mathbf{V}$, which on the one hand tend to predict as accurate as possible on training data, while on the other hand tend to explain the data well.

### 3.2 Optimization

**Variational approximation with Contrastive Divergence**: Since the data likelihood $L(\Theta)$ is generally intractable to compute, our method is based on the efficient contrastive divergence technique [16, 25, 26, 28]. Specifically, we derive a variational approximation $\mathcal{L}^v(q_0, q_1)$ of the negative log-likelihood $L(\Theta)$ , that is:

$$\mathcal{L}^v(q_0, q_1) := R(q_0(\mathbf{x}, \mathbf{z}, \mathbf{h}), p(\mathbf{x}, \mathbf{z}, \mathbf{h})) - R(q_1(\mathbf{x}, \mathbf{z}, \mathbf{h}), p(\mathbf{x}, \mathbf{z}, \mathbf{h})),$$

where $R(q, p)$ is the relative entropy, and $q_0$ is a variational distribution with $\mathbf{x}$ and $\mathbf{z}$ clamped to their observed values while $q_1$ is a distribution with all variables free. For $q$ ($q_0$ or $q_1$) in general, we make the *structured* mean field assumption [27] that [1] $q(\mathbf{x}, \mathbf{z}, \mathbf{h}) = q(\mathbf{x})q(\mathbf{z})q(\mathbf{h})$.

**Solving the approximate problem**: Applying the variational approximation $\mathcal{L}^v$ in problem (4), we get an approximate objective function $\mathcal{L}(\Theta, \mathbf{V}, q_0, q_1)$. Then, we can develop an alternating minimization method, which iteratively minimizes $\mathcal{L}(\Theta, \mathbf{V}, q_0, q_1)$ over $q_0$ and $(\Theta, \mathbf{V})$. The distribution $q_1$ is reconstructed once the optimal $q_0$ is achieved, see [25] for details.

The problem of solving $q_0$ and $q_1$ is the *posterior inference* problem. Specifically, for a variational distribution $q$ (can be $q_0$ or $q_1$) in general, we keep $(\Theta, \mathbf{V})$ fixed and update each marginal as

$$q(\mathbf{x}) = p(\mathbf{x}|\mathbb{E}_{q(\mathbf{H})}[\mathbf{H}]), \ \ q(\mathbf{z}) = p(\mathbf{z}|\mathbb{E}_{q(\mathbf{H})}[\mathbf{H}]), \ \text{ and } \ q(\mathbf{h}) = \prod_k p(h_k|\mathbb{E}_{q(\mathbf{X})}[\mathbf{X}], \mathbb{E}_{q(\mathbf{Z})}[\mathbf{Z}]). \ (5)$$

For $q_0$, $(\mathbf{x}, \mathbf{z})$ are clamped at their observed values, and only $q_0(\mathbf{h})$ is updated, which can be very efficiently done because of its factorized form. The distribution $q_1$ is achieved by performing the above updates starting from $q_0$. Several iterations can yield a good $q_1$. Again, we can see that both $q(\mathbf{x})$ and $q(\mathbf{z})$ are CRFs, with the expectation of $\mathbf{H}$ as the condition. Therefore, for linear-chain models, we can use a message passing scheme [18] to infer their marginal distributions, as needed for parameter estimation and view-level prediction (e.g., image annotation), as we shall see. For generally structured models, approximate inference techniques [22] can be applied.

After we have inferred $q_0$ and $q_1$, parameter estimation can be done by alternating between (1) estimating $\mathbf{V}$ with $\Theta$ fixed: this problem is learning a multi-class SVM [11], which can be

efficiently done with existing solvers; and (2) estimating $\Theta$ with $\mathbf{V}$ fixed: this can be solved with sub-gradient descent, where the sub-gradient is computed as:

$$\nabla\theta_i = -\mathbb{E}_{q_0}[\phi(x_i, x_{i+1})] + \mathbb{E}_{q_1}[\phi(x_i, x_{i+1})],$$
$$\nabla\eta_j = -\mathbb{E}_{q_0}[\psi(z_j, z_{j+1})] + \mathbb{E}_{q_1}[\psi(z_j, z_{j+1})],$$
$$\nabla\lambda_k = -\mathbb{E}_{q_0}[\varphi(h_k)] + \mathbb{E}_{q_1}[\varphi(h_k)],$$
$$\nabla\mathbf{W}_i^k = -\mathbb{E}_{q_0}[\phi(x_i, x_{i+1})\varphi(h_k)^\top] + \mathbb{E}_{q_1}[\phi(x_i, x_{i+1})\varphi(h_k)^\top] - C_2\tfrac{1}{D}\sum_d(\mathbf{V}_{\bar{y}_d k} - \mathbf{V}_{y_d k})\tfrac{\partial\mathbb{E}_{q_0}[h_k]}{\partial\mathbf{W}_i^k},$$
$$\nabla\mathbf{U}_j^k = -\mathbb{E}_{q_0}[\psi(z_j, z_{j+1})\varphi(h_k)^\top] + \mathbb{E}_{q_1}[\psi(z_j, z_{j+1})\varphi(h_k)^\top] - C_2\tfrac{1}{D}\sum_d(\mathbf{V}_{\bar{y}_d k} - \mathbf{V}_{y_d k})\tfrac{\partial\mathbb{E}_{q_0}[h_k]}{\partial\mathbf{U}_j^k},$$

where $\bar{y}_d = \arg\max_y[\Delta\ell_d(y) + \mathbf{V}^\top\mathbb{E}_{q_0}[\mathbf{f}(\mathbf{H}_d, y)]$ is the *loss-augmented prediction*, and the expectation $\mathbb{E}_{q_0}[\phi(x_i, x_{i+1})]$ is actually the count frequency of $\phi(x_i, x_{i+1})$, likewise for $\mathbb{E}_{q_0}[\psi(z_j, z_{j+1})]$.

Note that in our integrated max-margin formulation, the sub-gradients of $\mathbf{W}$ and $\mathbf{U}$ contain an additional term (i.e., the third term) compared to the standard DWH [28] with contrastive divergence approximation. This additional term introduces a regularization effect to the latent subspace model. If the prediction label $y_d$ differs from the true label $\bar{y}_d$, this term will be non-zero and it biases the model towards discovering a better representation for prediction.

## 4   Application to Image Classification, Annotation and Retrieval

We have developed the large-margin framework with a generic multi-view latent space MN to model structured data. In order to carefully examine the basic learning principle and compare with existing work, in this paper, we concentrate on a simplified but very rich case that the data on each view are not structured, which has been extensively studied in EFH [26, 28, 29] for image classification, annotation and retrieval. We denote the specialized model by MMH (max-margin Harmonium). In theory, extensions to model structured multi-view data can be easily done under the general framework, and the only needed change is on the step of inferring $q_1$, which can be treated as a black box, given the wide literature on approximate inference [22]. We defer the systematical study in this direction to the full extension of this work.

Specifically, we consider two-views, where $\mathbf{x}$ is a vector of discrete word features (e.g., image tags) and $\mathbf{z}$ is a vector of real-valued features (e.g., color histograms). Each $x_i$ is a Bernoulli variable that denotes whether the $i$th term of a dictionary appears or not in an image, and each $z_j$ is a real number that denotes the normalized color histogram of an image. We assume that each real-valued $h_k$ follows a univariate Gaussian distribution. Therefore, we define the conditional distributions as

$$p(x_i=1|\mathbf{h}) = \frac{1}{1 + e^{-(\alpha_i + \mathbf{W}_i.\mathbf{h})}}, \ p(z_j|\mathbf{h}) = \mathcal{N}(z_j|\sigma_j^2(\beta_j + \mathbf{U}_j.\mathbf{h}), \sigma_j^2), \ p(h_k|\mathbf{x}, \mathbf{z}) = \mathcal{N}(h_k|\mathbf{x}^\top\mathbf{W}_{.k} + \mathbf{z}^\top\mathbf{U}_{.k}, 1),$$

where $\mathbf{W}_i.$ and $\mathbf{W}_{.k}$ denote the $i$th row and $k$th column of $\mathbf{W}$, respectively. Alike for $\mathbf{U}_i.$ and $\mathbf{U}_{.k}$.

With the above definitions, we can follow exactly the same procedure as above to do parameter estimation. For the step of inferring $q_0$ and $q_1$, the distributions of $\mathbf{x}$, $\mathbf{z}$ and $\mathbf{h}$ are all fully factorized. Therefore, the sub-gradients can be easily computed. Details are deferred to the Appendix.

**Testing**: For classification and retrieval, we need to infer the posterior distribution of $\mathbf{H}$ and its expectation. In this case, we have $\mathbb{E}_{p(\mathbf{h}|\mathbf{x}, \mathbf{z})}[\mathbf{H}] = \mathbf{v}$, where $\mathbf{v}_k = \mathbf{x}^\top\mathbf{W}_{.k} + \mathbf{z}^\top\mathbf{U}_{.k}$, $\forall 1 \le k \le K$. Therefore, the *classification* rule is $y^* = \arg\max_y \mathbf{V}^\top\mathbf{f}(\mathbf{v}, y)$. For *retrieval*, the expectation $\mathbf{v}$ of each image is used to compute a similarity (e.g., cosine) between images. For *annotation*, we use $\mathbf{x}$ to represent tags, which are observed in training. In testing, we infer the posterior distribution $p(\mathbf{x}|\mathbf{z})$, which can be approximately computed by running the update equations (5) with $\mathbf{z}$ clamped at its observed values. Then, tags with high probabilities are selected as annotation.

## 5   Experiments

We report empirical results on TRECVID2003 and flickr image datasets. Our results demonstrate that the large-margin approach can achieve significantly better performance on discovering predictive subspace representations and the tasks of image classification, annotation and retrieval.

### 5.1   Datasets and Features

The first dataset is the TRECVID2003 video dataset [28], which contains 1078 manually labeled video shots that belong to 5 categories. Each shot is represented as a 1894-dim vector of text features

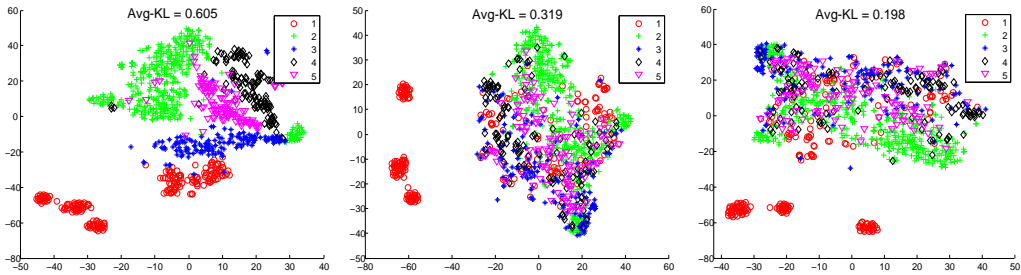

Figure 2: t-SNE 2D embedding of the discovered latent space representation by (Left) MMH, (Middle) DWH and (Right) TWH on the TRECVID video dataset (Better viewed in color).

and a 165-dim vector of HSV color histogram, which is extracted from the associated keyframe. We evenly split this dataset for training and testing. The second one is a subset selected from NUS-WIDE [10], which is a big image dataset constructed from flickr web images. This dataset contains 3411 images about 13 animals, including *cat*, *tiger*, etc. See Fig. 6 for example images for each category. For each image, six types of low-level features [10] are extracted, including 634-dim real valued features (i.e., 64-dim color histogram, 144-dim color correlogram, 73-dim edge direction histogram, 128-dim wavelet texture and 225-dim block-wise color moments) and 500-dim bag-of-word representation based on SIFT [19] features. We randomly select 2054 images for training and use the rest for testing. The online tags are also downloaded for evaluating image annotation.

## 5.2  Discovering Predictive Latent Subspace Representations

We first evaluate the predictive power of the discovered latent subspace representations.

Fig. 2 shows the 2D embedding of the discovered 10-dim latent representations by three models (i.e., MMH, DWH and TWH) on the video data. Here, we use the t-SNE algorithm [21] to find the embedding. We can see that clearly the latent subspace representations discovered by the large-margin based MMH show a strong grouping pattern for the images belonging to the same category, while images from different categories tend to be separated from each other on the 2D embedding space. In contrast, the latent subspace representations discovered by the likelihood-based unsupervised DWH and supervised TWH do not show a clear grouping pattern, except for the first category. Images from different categories tend to mix together. These observations suggest that the large-margin based latent subspace model can discover more predictive or discriminative latent subspace representations, which will result in better prediction performance, as we shall see.

To quantitatively evaluate the predictiveness of the discovered latent subspace representations, we compute the pair-wise average KL-divergence between the per-class average distribution over latent topics[2]. As shown on the top of each plot in Fig. 2, the large-margin based MMH obtains a much larger average KL-divergence than the other likelihood-based methods. This again suggests that the latent subspace representations discovered by MMH are more discriminative or predictive. We obtain the similar observations and conclusions on the flickr dataset (see Fig. 3 for some example topics), where the average KL-divergence scores of 60-topic MMH, DWH and TWH are 3.23, 2.56 and 0.463, respectively.

Finally, we examine the predictive power of discovered latent topics. Fig. 3 shows five example topics discovered by the large-margin MMH on the flickr image data. For each topic $H_k$, we show the 5 top-ranked images that yield a high expected value of $H_k$, together with the associated tags. Also, to qualitatively visualize the discriminative power of each topic among the 13 categories, we show the average probability of each category distributed on the particular topic. From the results, we can see that many of the discovered topics are very predictive for one or several categories. For example, topics 3 and 4 are discriminative in predicting the categories *hawk* and *whales*, respectively. Similarly, topics 1 and 5 are good at predicting *squirrel* and *zebra*, respectively. We also have some topics which are good at discriminating a subset of categories against another subset. For example, the topic 2 is good at discriminating {*squirrel*, *wolf*, *rabbit*} against {*tiger*, *whales*, *zebra*}; but it is not very discriminative between *squirrel* and *wolf*.

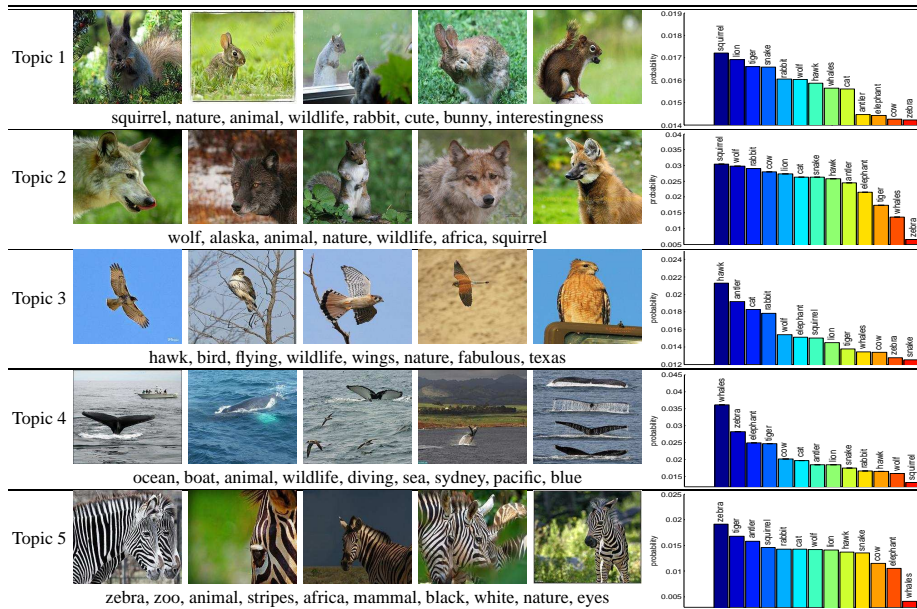

Figure 3: Example latent topics discovered by a 60-topic MMH on the flickr animal dataset.

## 5.3 Prediction Performance on Image Classification, Retrieval, and Annotation

### 5.3.1 Classification

We first compare the MMH with SVM, DWH, TWH, Gaussian Mixture (GM-Mix), Gaussian Mixture LDA (GM-LDA), and Correspondence LDA (CorrLDA) on the TRECVID data. See [4] for the details of the last three models. We use the $SVM^{struct}$ [3] to solve the sub-step of learning $\mathbf{V}$ in MMH and build an SVM classifier, which uses both the text and color histogram features without distinguishing them in different views. For each of the unsupervised DWH, GM-Mix, GM-LDA and CorrLDA, a downstream SVM is built with the same tool based on the discovered latent representations. Fig. 4 (a) shows the classification accuracy of different models, where CorrLDA is omitted because of its too low performance. We can see that the max-margin based multi-view MMH performs consistently better than any other competitors. In contrast, the likelihood-based TWH does not show any conclusive improvements compared to the unsupervised DWH. These results show that supervised information can help in discovering predictive latent space representations that are more suitable for prediction if the model is appropriately learned, e.g., by using the large-margin method. The superior performance of MMH compared to the flat SVM demonstrates the usefulness of modeling multi-view inputs for prediction. The reasons for the inferior performance of other models (e.g., CorrLDA and GM-Mix) are analyzed in [28, 29].

Fig. 4 (b) shows the classification accuracy on the flickr animal dataset. For brevity, we compare MMH only with the best performed DWH, TWH and SVM. For these methods, we use the 500-dim SIFT and 634-dim real features, which are treated as two views of inputs for MMH, DWH and TWH. Also, we compare with the single-view MedLDA [31], which uses SIFT features only. To be fair, we also evaluate a version of MMH that uses SIFT features, and denote it by MMH (SIFT). Again, we can see that the large-margin based multi-view MMH performs much better than any other methods, including SVM which ignores the presence of multi-view features. For the single-view MMH (SIFT), it performs comparably (slightly better than) with the large-margin based MedLDA, which is a directed BN. With the similar large-margin principle, MMH is an important extension of MedLDA to the undirected latent subspace models and for multi-view data analysis.

### 5.3.2 Retrieval

For image retrieval, each test image is treated as a query and training images are ranked based on their cosine similarity with the given query, which is computed based on latent subspace representations. An image is considered relevant to the query if they belong to the same category. We evaluate the retrieval results by computing the average precision (AP) score and drawing precision-recall curves. Fig. 4 (c) compares MMH with four other models when the topic number changes. Here,

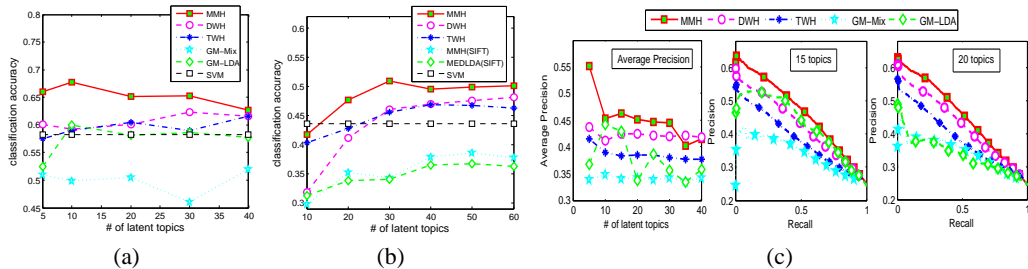

Figure 4: Classification accuracy on the (a) TRECVID 2003 and (b) flickr datasets and (c) the average precision curve and the two precision-recall curves for image retrieval on TRECVID data.

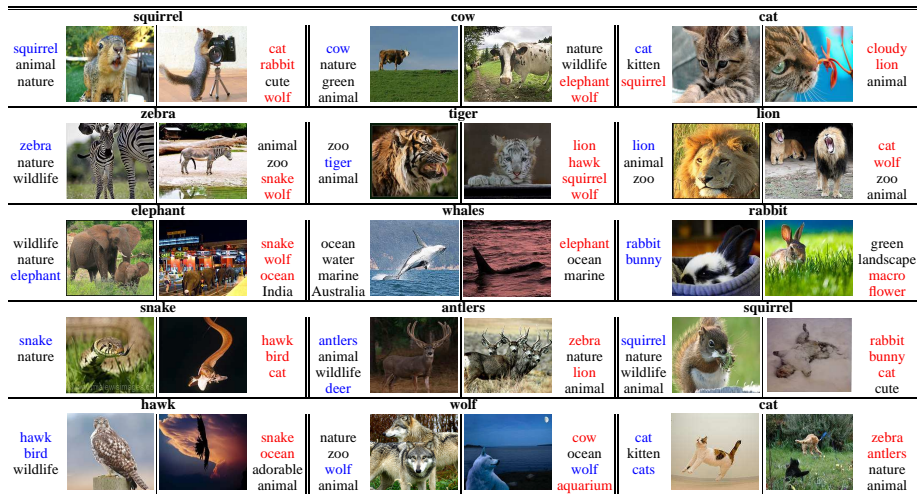

Figure 6: Example images from the 13 categories on the flickr animal dataset with predicted annotations. Tags in blue are correct annotations while red ones are wrong predictions. The other tags are neutral.

we show the precision-recall curves when the topic number is set at 15 and 20. We can see that for the AP measure, MMH outperforms all other methods in most cases, and MMH consistently outperforms all the other methods in the measure of precision-recall curve. On the flickr dataset, we have similar observations. The AP scores of the 60-topic MMH, DWH, and TWH are 0.163, 0.153 and 0.158, respectively. Due to space limitation, we defer the details to a full extension.

### 5.3.3 Annotation

Finally, we report the annotation results on the flickr dataset, with a dictionary of 1000 unique tags. The average number of tags per image is about $4.5$. We compare MMH with DWH and TWH with two views of inputs–$\mathbf{X}$ for tag and $\mathbf{Z}$ for all the 634-dim real-valued features. We also compare with the sLDA annotation model [24], which uses SIFT features and tags as inputs. We use

|        | MMH    | DWH   | TWH   | sLDA  |
|--------|--------|-------|-------|-------|
| $F1@1$ | **0.165** | 0.144 | 0.145 | 0.077 |
| $F1@2$ | **0.221** | 0.186 | 0.192 | 0.124 |
| $F1@3$ | **0.245** | 0.202 | 0.218 | 0.146 |
| $F1@4$ | **0.258** | 0.208 | 0.228 | 0.159 |
| $F1@5$ | **0.262** | 0.210 | 0.236 | 0.169 |
| $F1@6$ | **0.259** | 0.208 | 0.240 | 0.171 |
| $F1@7$ | **0.256** | 0.206 | 0.239 | 0.175 |

Figure 5: Top-$N$ F1-measure.

the top-$N$ F1-measure [24], denoted by $F1@N$. With 60 latent topics, the top-$N$ F-measure scores are shown in Fig. 5. We can see that the large-margin based MMH significantly outperforms all the competitors. Fig. 6 shows example images from all the 13 categories, where for each category the left image is generally of a good annotation quality and the right one is relatively worse.

## 6 Conclusions and Future Work

We have presented a generic large-margin learning framework for discovering predictive latent subspace representations shared by structured multi-view data. The inference and learning can be efficiently done with contrastive divergence methods. Finally, we concentrate on a specialized model with applications to image classification, annotation and retrieval. Extensive experiments on real video and web image datasets demonstrate the advantages of large-margin learning for both prediction and predictive latent subspace discovery. In future work, we plan to systematically investigate the large-margin learning framework on structured multi-view data analysis, e.g., on text mining [23] and computer vision [15] applications.

## Acknowledgments

This work was done while N. Chen was a visiting researcher at CMU under a CSC fellowship and supports from Chinese NSF Grants (No. 60625304, 90716021, 61075027), the National Key Project for Basic Research of China (Grants No. G2007CB311003, 2009CB724002). J. Zhu and E. P. Xing are supported by ONR N000140910758, NSF IIS-0713379, NSF Career DBI-0546594, and an Alfred P. Sloan Research Fellowship.

## Footnotes

[1] The parametric form assumptions of $q$, as made in previous work [28, 29], are not needed.

[2]To compute this score, we first turn the expected value of $\mathbf{H}$ to be non-negative by subtracting each element by the smallest value and then normalize it into a distribution over the $K$ topics. The per-class average is computed by averaging the topic distributions of the images within the same class. For a pair of distributions $p$ and $q$, the average KL-divergence is $1/2(R(p,q) + R(q,p))$.

[3]http://svmlight.joachims.org/svm_multiclass.html

## References

[1] S. Akaho. A kernel method for canonical correlation analysis. In *IMPS*, 2001.

[2] K. Ando and T. Zhang. Two-view feature generation model for semi-supervised learning. In *ICML*, 2007.

[3] F. R. Bach and M. I. Jordan. A probabilistic interpretation of canonical correlation analysis. Technical report, Technical Report 688, Dept. of Statistics. University of California, 2005.

[4] D. M. Blei and M. I. Jordan. Modeling annotated data. In *ACM SIGIR*, pages 127–134, 2003.

[5] D. M. Blei, A. Y. Ng, and M. I. Jordan. Latent dirichlet allocation. *JMLR*, 3:993–1022, 2003.

[6] A. Blum and T. Mitchell. Combining labeled and unlabeled data with co-trainnig. In *COLT*, 1998.

[7] U. Brefeld and T. Scheffer. Co-EM support vector learning. In *ICML*, 2004.

[8] K. Chaudhuri, S. M. Kakade, K. Livescu, and K. Sridharan. Multi-view clustering via canonical correlation analysis. In *ICML*, 2009.

[9] C. M. Christoudias, R. Urtasun, and T. Darrell. Multi-view learning in the presence of view disagreement. In *UAI*, 2008.

[10] T.-S. Chua, J. Tang, R. Hong, H. Li, Z. Luo, and Y.-T. Zheng. NUS-WIDE: A real-world web image database from national university of singapore. In *CIVR*, 2009.

[11] K. Crammer and Y. Singer. On the algorithmic implementation of multiclass kernel-based vector machines. *JMLR*, (2):265–292, 2001.

[12] M. Culp, G. Michailidis, and K. Johnson. On multi-view learning with additive models. *Annals of Applied Statistics*, 3(1):292–318, 2009.

[13] T. Diethe, D. R. Hardoon, and J. Shawe-Taylor. Multiview fisher discriminant analysis. In *NIPS Workshop on Learning from Multiple Sources*, 2008.

[14] D. Foster, S. Kakade, and T. Zhang. Multi-view dimensionality reduction via canonical correlation analysis. Technical report, Technical Report TR-2008-4, TTI-Chicago, 2008.

[15] D. Gökalp and S. Aksoy. Scene classification using bag-of-regions representations. In *CVPR*, 2007.

[16] G. E. Hinton. Training products of experts by minimizing contrastive divergence. *Neural Computation*, 14(8):1771–1800, 2002.

[17] H. Hotelling. Relations between two sets of variates. *Biometrika*, 28(3/4):321–377, 1936.

[18] J. Lafferty, A. McCallum, and F. Pereira. Conditional random fields: Probabilistic models for segmenting and labeling sequence data. In *ICML*, 2001.

[19] D. G. Lowe. Object recognition from local scale-invariant features. In *CVPR*, 1999.

[20] R. Salakhutdinov and G. E. Hinton. Replicated softmax: an undirected topic model. In *NIPS*, 2009.

[21] L. van der Maaten and G. Hinton. Visualizing data using t-SNE. *JMLR*, 9:2579–2605, 2008.

[22] M. J. Wainwright and M. I. Jordan. Graphical models, exponential families, and variational inference. *Foundations and Trends in Machine Learning*, 1(1–2):1–305, 2008.

[23] H. M. Wallach. Topic modeling: Beyond bag-of-words. In *ICML*, 2006.

[24] C. Wang, D. M. Blei, and L. Fei-Fei. Simultaneous image classification and annotation. In *CVPR*, 2009.

[25] M. Welling and G. E. Hinton. A new learning algorithm for mean field boltzmann machines. In *ICANN*, 2001.

[26] M. Welling, M. Rosen-Zvi, and G. E. Hinton. Exponential family harmoniums with an application to information retrieval. In *NIPS*, pages 1481–1488, 2004.

[27] E. P. Xing, M. I. Jordan, and S. Russell. A generalized mean field algorithm for variational inference in exponential families. In *UAI*, 2003.

[28] E. P. Xing, R. Yan, and A. G. Hauptmann. Mining associated text and images with dual-wing harmoniums. In *UAI*, 2005.

[29] J. Yang, Y. Liu, E. P. Xing, and A. G. Hauptmann. Harmonium models for semantic video representation and classification. In *SDM*, 2007.

[30] J. Zhang, Z. Ghahramani, and Y. Yang. Flexible latent variable models for multi-task learning. *Machine Learning*, 73(3):221–242, 2008.

[31] J. Zhu, A. Ahmed, and E. P. Xing. MedLDA: Maximum margin supervised topic models for regression and classification. In *ICML*, 2009.

